# Predictive Q-Routing: A Memory-based Reinforcement Learning Approach to Adaptive Traffic Control

**Samuel P.M. Choi, Dit-Yan Yeung**
Department of Computer Science
Hong Kong University of Science and Technology
Clear Water Bay, Kowloon, Hong Kong
{pmchoi,dyyeung}@cs.ust.hk

## Abstract

In this paper, we propose a memory-based Q-learning algorithm called *predictive Q-routing* (PQ-routing) for adaptive traffic control. We attempt to address two problems encountered in Q-routing (Boyan & Littman, 1994), namely, the inability to fine-tune routing policies under low network load and the inability to learn new optimal policies under decreasing load conditions. Unlike other memory-based reinforcement learning algorithms in which memory is used to keep past experiences to increase learning speed, PQ-routing keeps the best experiences learned and reuses them by predicting the traffic trend. The effectiveness of PQ-routing has been verified under various network topologies and traffic conditions. Simulation results show that PQ-routing is superior to Q-routing in terms of both learning speed and adaptability.

## 1 INTRODUCTION

The adaptive traffic control problem is to devise routing policies for controllers (i.e. routers) operating in a non-stationary environment to minimize the average packet delivery time. The controllers usually have no or only very little prior knowledge of the environment. While only local communication between controllers is allowed, the controllers must cooperate among themselves to achieve the common, global objective. Finding the optimal routing policy in such a distributed manner is very difficult. Moreover, since the environment is non-stationary, the optimal policy varies with time as a result of changes in network traffic and topology.

In (Boyan & Littman, 1994), a distributed adaptive traffic control scheme based on reinforcement learning (RL), called *Q-routing*, is proposed for the routing of packets in networks with dynamically changing traffic and topology. Q-routing is a variant of Q-learning (Watkins, 1989), which is an incremental (or asynchronous) version of dynamic programming for solving multistage decision problems. Unlike the original Q-learning algorithm, Q-routing is distributed in the sense that each communication node has a separate local controller, which does not rely on global information of the network for decision making and refinement of its routing policy.

## 2   EXPLORATION VERSUS EXPLOITATION

As in other RL algorithms, one important issue Q-routing must deal with is the tradeoff between exploration and exploitation. While exploration of the state space is essential to learning good routing policies, continual exploration without putting the learned knowledge into practice is of no use. Moreover, exploration is not done at no cost. This dilemma is well known in the RL community and has been studied by some researchers, e.g. (Thrun, 1992).

One possibility is to divide learning into an exploration phase and an exploitation phase. The simplest exploration strategy is random exploration, in which actions are selected randomly without taking the reinforcement feedback into consideration. After the exploration phase, the optimal routing policy is simply to choose the next network node with minimum Q-value (i.e. minimum estimated delivery time). In so doing, Q-routing is expected to learn to avoid congestion along popular paths.

Although Q-routing is able to alleviate congestion along popular paths by routing some traffic over other (possibly longer) paths, two problems are reported in (Boyan & Littman, 1994). First, Q-routing is not always able to find the shortest paths under low network load. For example, if there exists a longer path which has a Q-value less than the (erroneous) estimate of the shortest path, a routing policy that acts as a minimum selector will not explore the shortest path and hence will not update its erroneous Q-value. Second, Q-routing suffers from the so-called hysteresis problem, in that it fails to adapt to the optimal (shortest) path again when the network load is lowered. Once a longer path is selected due to increase in network load, a minimum selector is no longer able to notice the subsequent decrease in traffic along the shortest path. Q-routing continues to choose the same (longer) path unless it also becomes congested and has a Q-value greater than some other path. Unless Q-routing continues to explore, the shortest path cannot be chosen again even though the network load has returned to a very low level. However, as mentioned in (Boyan & Littman, 1994), random exploration may have very negative effects on congestion, since packets sent along a suboptimal path tend to increase queue delays, slowing down all the packets passing through this path.

Instead of having two separate phases for exploration and exploitation, one alternative is to mix them together, with the emphasis shifting gradually from the former to the latter as learning proceeds. This can be achieved by a probabilistic scheme for choosing next nodes. For example, the Q-values may be related to probabilities by the Boltzmann-Gibbs distribution, involving a randomness (or pseudo-temperature) parameter $T$. To guarantee sufficient initial exploration and subsequent convergence, $T$ usually has a large initial value (giving a uniform probability distribution) and decreases towards 0 (degenerating to a deterministic minimum selector) during the learning process. However, for a continuously operating network with dynamically changing traffic and topology, learning must be continual and hence cannot be controlled by a prespecified decay profile for $T$. An algorithm which automatically adapts between exploration and exploitation is therefore necessary. It is this very reason which led us to develop the algorithm presented in this paper.

## 3   PREDICTIVE Q-ROUTING

A memory-based Q-learning algorithm called *predictive Q-routing* (PQ-routing) is proposed here for adaptive traffic control. Unlike *Dyna* (Peng & Williams, 1993) and *prioritized sweeping* (Moore & Atkeson, 1993) in which memory is used to keep past experiences to increase learning speed, PQ-routing keeps the best experiences (best Q-values) learned and reuses them by predicting the traffic trend. The idea is as follows. Under low network load, the optimal policy is simply the shortest path routing policy. However, when the load level increases, packets tend to queue up along the shortest paths and the simple shortest path routing policy no longer performs well. If the congested paths are not used for a period of time, they will recover and become good candidates again. One should therefore try to utilize these paths by occasionally sending packets along them. We refer to such controlled exploration activities as *probing*. The probing frequency is crucial, as frequent probes will increase the load level along the already congested paths while infrequent probes will make the performance little different from Q-routing. Intuitively, the probing frequency should depend on the congestion level and the processing speed (recovery rate) of a path. The congestion level can be reflected by the current Q-value, but the recovery rate has to be estimated as part of the learning process.

At first glance, it seems that the recovery rate can be computed simply by dividing the difference in Q-values from two probes by the elapse time. However, the recovery rate changes over time and depends on the current network traffic and the possibility of link/node failure. In addition, the elapse time does not truly reflect the actual processing time a path needs. Thus this noisy recovery rate should be adjusted for every packet sent. It is important to note that the recovery rate in the algorithm should not be positive, otherwise it may increase the predicted Q-value without bound and hence the path can never be used again.

### Predictive Q-Routing Algorithm

TABLES:
$Q_x(d,y)$ - estimated delivery time from node $x$ to node $d$ via neighboring node $y$
$B_x(d,y)$ - best estimated delivery time from node $x$ to node $d$ via neighboring node $y$
$R_x(d,y)$ - recovery rate for path from node $x$ to node $d$ via neighboring node $y$
$U_x(d,y)$ - last update time for path from node $x$ to node $d$ via neighboring node $y$

TABLE UPDATES: (after a packet arrives at node $y$ from node $x$)
$\Delta Q = $ (transmission delay + queueing time at $y$ + $\min_z\{Q_y(d,z)\}$) $- Q_x(d,y)$
$Q_x(d,y) \leftarrow Q_x(d,y) + \alpha\,\Delta Q$
$B_x(d,y) \leftarrow \min(B_x(d,y), Q_x(d,y))$
if $(\Delta Q < 0)$ then
    $\Delta R \leftarrow \Delta Q\,/\,$(current time $- U_x(d,y)$)
    $R_x(d,y) \leftarrow R_x(d,y) + \beta\,\Delta R$
else if $(\Delta Q > 0)$ then
    $R_x(d,y) \leftarrow \gamma\,R_x(d,y)$
end if
$U_x(d,y) \leftarrow$ current time

ROUTING POLICY: (packet is sent from node $x$ to node $y$)
$\Delta t = $ current time $- U_x(d,y)$
$Q'_x(d,y) = \max(Q_x(d,y) + \Delta t\,R_x(d,y), B_x(d,y))$
$y \leftarrow \arg\min_y\{Q'_x(d,y)\}$

There are three learning parameters in the PQ-routing algorithm. $\alpha$ is the Q-function learning parameter as in the original Q-learning algorithm. In PQ-routing, this parameter should be set to 1 or else the accuracy of the recovery rate may be

affected. $\beta$ is used for learning the recovery rate. In our experiments, the value of 0.7 is used. $\gamma$ is used for controlling the decay of the recovery rate, which affects the probing frequency in a congested path. Its value is usually chosen to be larger than $\beta$. In our experiments, the value of 0.9 is used.

PQ-learning is identical to Q-learning in the way the Q-function is updated. The major difference is in the routing policy. Instead of selecting actions based solely on the current Q-values, the recovery rates are used to yield better estimates of the Q-values before the minimum selector is applied. This is desirable because the Q-values on which routing decisions are based may become outdated due to the ever-changing traffic.

# 4   EMPIRICAL RESULTS

## 4.1   A 15-NODE NETWORK

To demonstrate the effectiveness of PQ-routing, let us first consider a simple 15-node network (Figure 1(a)) with three sources (nodes 12 to 14) and one destination (node 15). Each node can process one packet per time step, except nodes 7 to 11 which are two times faster than the other nodes. Each link is bidirectional and has a transmission delay of one time unit. It is not difficult to see that the shortest paths are $12 \rightarrow 1 \rightarrow 4 \rightarrow 15$ for node 12, $13 \rightarrow 2 \rightarrow 4 \rightarrow 15$ for node 13, and $14 \rightarrow 3 \rightarrow 4 \rightarrow 15$ for node 14. However, since each node along these paths can process only one packet per time step, congestion will soon occur in node 4 if all source nodes send packets along the shortest paths.

One solution to this problem is that the source nodes send packets along different paths which share no common nodes. For instance, node 12 can send packets along path $12 \rightarrow 1 \rightarrow 5 \rightarrow 6 \rightarrow 15$ while node 13 along $13 \rightarrow 2 \rightarrow 7 \rightarrow 8 \rightarrow 9 \rightarrow 10 \rightarrow 11 \rightarrow 15$ and node 14 along $14 \rightarrow 3 \rightarrow 4 \rightarrow 15$. The optimal routing policy depends on the traffic from each source node. If the network load is not too high, the optimal routing policy is to alternate between the upper and middle paths in sending packets.

## 4.1.1   PERIODIC TRAFFIC PATTERNS UNDER LOW LOAD

For the convenience of empirical analysis, we first consider periodic traffic in which each source node generates the same traffic pattern over a period of time. Figure 1(b) shows the average delivery time for Q-routing and PQ-routing. PQ-routing performs better than Q-routing after the initial exploration phase (25 time steps), despite of some slight oscillations. Such oscillations are due to the occasional probing activities of the algorithm. When we examine Q-routing more closely, we can find that after the initial learning, all the source nodes try to send packets along the upper (shortest) path, leading to congestion in node 4. When this occurs, both nodes 12 and 13 switch to the middle path, which subsequently leads to congestion in node 5. Later, nodes 12 and 13 detect this congestion and then switch to the lower path. Since the nodes along this path have higher (two times) processing speed, the Q-values become stable and Q-routing will stay there as long as the load level does not increase. Thus, Q-routing fails to fine-tune the routing policy to improve it. PQ-routing, on the other hand, is able to learn the recovery rates and alternate between the upper and middle paths.

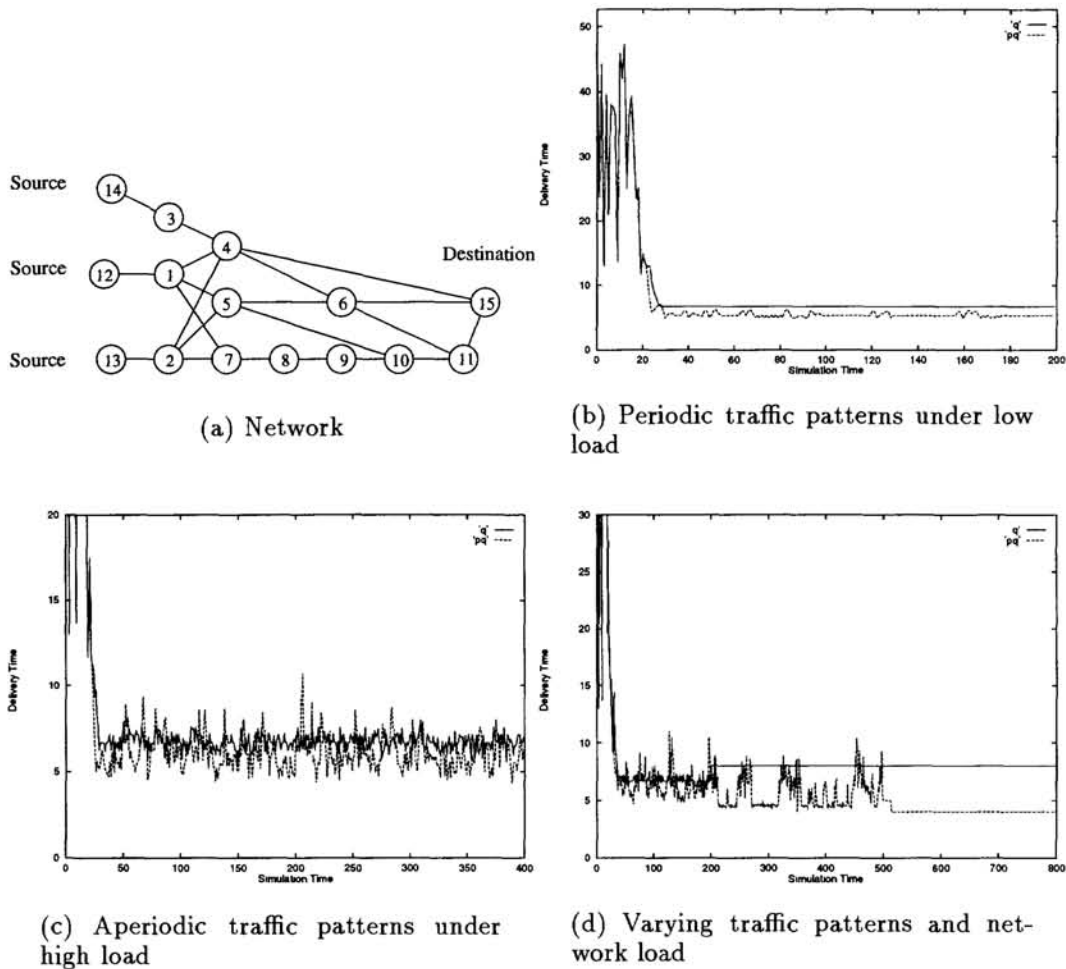

(a) Network

(b) Periodic traffic patterns under low load

(c) Aperiodic traffic patterns under high load

(d) Varying traffic patterns and network load

Figure 1: A 15-Node Network and Simulation Results

## 4.1.2   APERIODIC TRAFFIC PATTERNS UNDER HIGH LOAD

It is not realistic to assume that network traffic is strictly periodic. In reality, the time interval between two packets sent by a node varies. To simulate varying intervals between packets, a probability of 0.8 is imposed on each source node for generating packets. In this case, the average delivery time for both algorithms oscillates. Figure 1(c) shows the performance of Q-routing and PQ-routing under high network load. The difference in delivery time between Q-routing and PQ-routing becomes less significant, as there is less available bandwidth in the shortest path for interleaving. Nevertheless, it can be seen that the overall performance of PQ-routing is still better than Q-routing.

## 4.1.3   VARYING TRAFFIC PATTERNS AND NETWORK LOAD

In the more complicated situation of varying traffic patterns and network load, PQ-routing also performs better than Q-routing. Figure 1(d) shows the hysteresis problem in Q-routing under gradually changing traffic patterns and network load. After an initial exploration phase of 25 time steps, the load level is set to medium

from time step 26 to 200. From step 201 to 500, node 14 ceases to send packets and nodes 12 and 13 slightly increase their load level. In this case, although the shortest path becomes available again, Q-routing is not able to notice the change in traffic and still uses the same routing policy, but PQ-routing is able to utilize the optimal paths. After step 500, node 13 also ceases to send packets. PQ-routing is successful in adapting to the optimal path $12 \rightarrow 1 \rightarrow 4 \rightarrow 15$.

## 4.2   A 6x6 GRID NETWORK

Experiments have been performed on some larger networks, including a 32-node hypercube and some random networks, with results similar to those above. Figures 2(b) and 2(c) depict results for Boyan and Littman's 6x6 grid network (Figure 2(a)) under varying traffic patterns and network load.

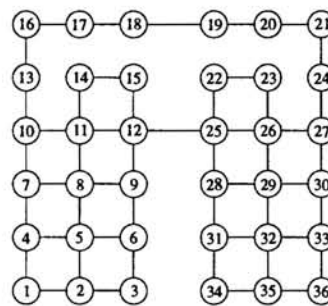

(a) Network

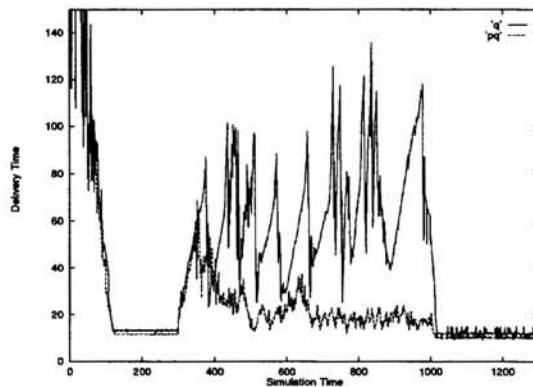

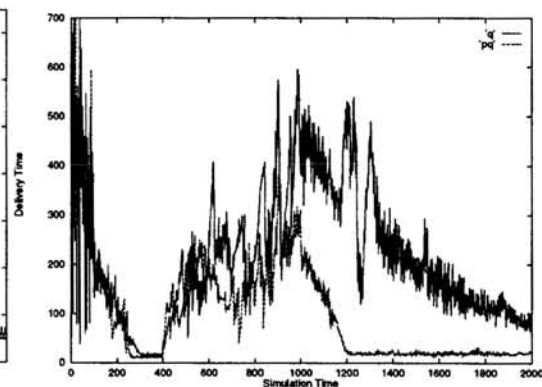

(b) Varying traffic patterns and network load

(c) Varying traffic patterns and network load

Figure 2: A 6x6 Grid Network and Simulation Results

In Figure 2(b), after an initial exploration for 50 time steps, the load level is set to low. From step 51 to 300, the load level increases to medium but with the same periodic traffic patterns. PQ-routing performs slightly better. From step 301 to 1000, the traffic patterns change dramatically under high network load. Q-routing cannot learn a stable policy in this (short) period of time, but PQ-routing becomes more stable after about 200 steps. From step 1000 onwards, the traffic patterns change again and the load level returns to low. PQ-routing still performs better.

In Figure 2(c), the first 100 time steps are for initial exploration. After this period, packets are sent from the bottom right part of the grid to the bottom left part with low network load. PQ-routing is found to be as good as the shortest path routing policy, while Q-routing is slightly poorer than PQ-routing. From step 400 to 1000, packets are sent from both the left and right parts of the grid to the opposite sides at high load level. Both the two bottleneck paths become congested and hence the average delivery time increases for both algorithms. From time step 1000 onwards, the network load decreases to a more manageable level. We can see that PQ-routing is faster than Q-routing in adapting to this change.

## 5    DISCUSSIONS

PQ-learning is generally better than Q-learning under both low and varying network load conditions. Under high load conditions, they give comparable performance. In general, Q-routing prefers stable routing policies and tends to send packets along paths with higher processing power, regardless of the actual packet delivery time. This strategy is good under extremely high load conditions, but may not be optimal under other situations. PQ-routing, on the contrary, is more aggressive. It tries to minimize the average delivery time by occasionally probing the shortest paths. If the load level remains extremely high with the patterns unchanged, PQ-routing will gradually degenerate to Q-routing, until the traffic changes again. Another advantage PQ-routing has over Q-routing is that shorter adaptation time is generally needed when the traffic patterns change, since the routing policy of PQ-routing depends not only on the current Q-values but also on the recovery rates. In terms of memory requirement, PQ-routing needs more memory for recovery rate estimation. It should be noted, however, that extra memory is needed only for the visited states. In the worst case, it is still in the same order as that of the original Q-routing algorithm. In terms of computational cost, recovery rate estimation is computationally quite simple. Thus the overhead for implementing PQ-routing should be minimal.

### References

J.A. Boyan & M.L. Littman (1994). Packet routing in dynamically changing networks: a reinforcement learning approach. *Advances in Neural Information Processing Systems 6*, 671–678. Morgan Kaufmann, San Mateo, California.

M. Littman & J. Boyan (1993). A distributed reinforcement learning scheme for network routing. *Proceedings of the First International Workshop on Applications of Neural Networks to Telecommunications*, 45–51. Lawrence Erlbaum, Hillsdale, New Jersey.

A.W. Moore & C.G. Atkeson (1993). Memory-based reinforcement learning: efficient computation with prioritized sweeping. *Advances in Neural Information Processing Systems 5*, 263-270. Morgan Kaufmann, San Mateo, California.

A.W. Moore & C.G. Atkeson (1993). Prioritized sweeping: reinforcement learning with less data and less time. *Machine Learning*, 13:103–130.

J. Peng & R.J. Williams (1993). Efficient learning and planning within the Dyna framework. *Adaptive Behavior*, 1:437–454.

S. Thrun (1992). The role of exploration in learning control. In *Handbook of Intelligent Control: Neural, Fuzzy, and Adaptive Approaches*, D.A. White & D.A. Sofge (eds). Van Nostrand Reinhold, New York.

C.J.C.H. Watkins (1989). *Learning from delayed rewards*. PhD Thesis, University of Cambridge, England.
